# Learning with Local and Global Consistency

**Dengyong Zhou, Olivier Bousquet, Thomas Navin Lal,**
**Jason Weston, and Bernhard Schölkopf**
Max Planck Institute for Biological Cybernetics, 72076 Tuebingen, Germany
{*firstname.secondname*}@*tuebingen.mpg.de*

## Abstract

We consider the general problem of learning from labeled and unlabeled data, which is often called semi-supervised learning or transductive inference. A principled approach to semi-supervised learning is to design a classifying function which is sufficiently *smooth* with respect to the intrinsic structure collectively revealed by known labeled and unlabeled points. We present a simple algorithm to obtain such a smooth solution. Our method yields encouraging experimental results on a number of classification problems and demonstrates effective use of unlabeled data.

## 1 Introduction

We consider the general problem of learning from labeled and unlabeled data. Given a point set $\mathcal{X} = \{x_1, \ldots, x_l, x_{l+1}, \ldots, x_n\}$ and a label set $\mathcal{L} = \{1, \ldots, c\}$, the first $l$ points have labels $\{y_1, \ldots, y_l\} \in \mathcal{L}$ and the remaining points are unlabeled. The goal is to predict the labels of the unlabeled points. The performance of an algorithm is measured by the error rate on these unlabeled points only.

Such a learning problem is often called semi-supervised or transductive. Since labeling often requires expensive human labor, whereas unlabeled data is far easier to obtain, semi-supervised learning is very useful in many real-world problems and has recently attracted a considerable amount of research [10]. A typical application is web categorization, in which manually classified web pages are always a very small part of the entire web, and the number of unlabeled examples is large.

The key to semi-supervised learning problems is the prior assumption of consistency, which means: (1) nearby points are likely to have the same label; and (2) points on the same structure (typically referred to as a cluster or a manifold) are likely to have the same label. This argument is akin to that in [2, 3, 4, 10, 15] and often called the *cluster assumption* [4, 10]. Note that the first assumption is local, whereas the second one is global. Orthodox supervised learning algorithms, such as $k$-NN, in general depend only on the first assumption of local consistency.

To illustrate the prior assumption of consistency underlying semi-supervised learning, let us consider a toy dataset generated according to a pattern of two intertwining moons in Figure 1(a). Every point should be similar to points in its local neighborhood, and furthermore, points in one moon should be more similar to each other than to points in the other moon. The classification results given by the Support Vector Machine (SVM) with a RBF kernel

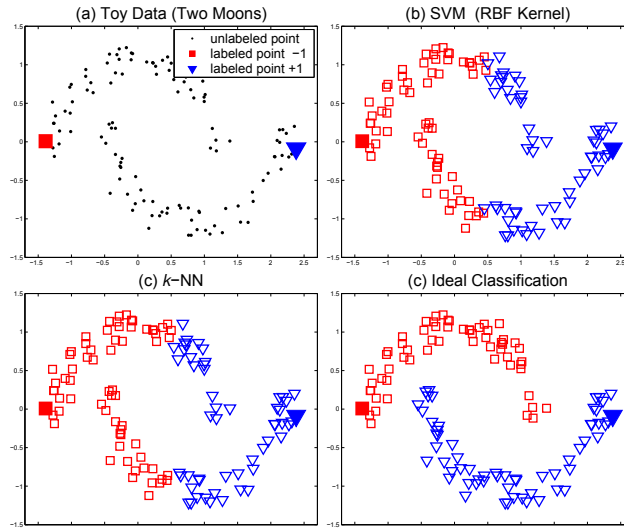

Figure 1: Classification on the two moons pattern. (a) toy data set with two labeled points; (b) classifying result given by the SVM with a RBF kernel; (c) $k$-NN with $k = 1$; (d) ideal classification that we hope to obtain.

and $k$-NN are shown in Figure 1(b) & 1(c) respectively. According to the assumption of consistency, however, the two moons should be classified as shown in Figure 1(d).

The main differences between the various semi-supervised learning algorithms, such as spectral methods [2, 4, 6], random walks [13, 15], graph mincuts [3] and transductive SVM [14], lie in their way of realizing the assumption of consistency. A principled approach to formalize the assumption is to design a classifying function which is sufficiently *smooth* with respect to the intrinsic structure revealed by known labeled and unlabeled points. Here we propose a simple iteration algorithm to construct such a smooth function inspired by the work on spreading activation networks [1, 11] and diffusion kernels [7, 8, 12], recent work on semi-supervised learning and clustering [2, 4, 9], and more specifically by the work of Zhu *et al.* [15]. The keynote of our method is to let every point iteratively spread its label information to its neighbors until a global stable state is achieved.

We organize the paper as follows: Section 2 shows the algorithm in detail and also discusses possible variants; Section 3 introduces a regularization framework for the method; Section 4 presents the experimental results for toy data, digit recognition and text classification, and Section 5 concludes this paper and points out the next researches.

## 2 Algorithm

Given a point set $\mathcal{X} = \{x_1, \ldots, x_l, x_{l+1}, \ldots, x_n\} \subset \mathbb{R}^m$ and a label set $\mathcal{L} = \{1, \ldots, c\}$, the first $l$ points $x_i (i \leq l)$ are labeled as $y_i \in \mathcal{L}$ and the remaining points $x_u (l+1 \leq u \leq n)$ are unlabeled. The goal is to predict the label of the unlabeled points.

Let $\mathcal{F}$ denote the set of $n \times c$ matrices with nonnegative entries. A matrix $F = [F_1^T, \ldots, F_n^T]^T \in \mathcal{F}$ corresponds to a classification on the dataset $\mathcal{X}$ by labeling each point $x_i$ as a label $y_i = \arg \max_{j \leq c} F_{ij}$. We can understand $F$ as a vectorial function $F : \mathcal{X} \to \mathbb{R}^c$ which assigns a vector $F_i$ to each point $x_i$. Define a $n \times c$ matrix $Y \in \mathcal{F}$ with $Y_{ij} = 1$ if $x_i$ is labeled as $y_i = j$ and $Y_{ij} = 0$ otherwise. Clearly, $Y$ is consistent with the

initial labels according the decision rule. The algorithm is as follows:

1. Form the affinity matrix $W$ defined by $W_{ij} = \exp(-\|x_i - x_j\|^2/2\sigma^2)$ if $i \neq j$ and $W_{ii} = 0$.

2. Construct the matrix $S = D^{-1/2}WD^{-1/2}$ in which $D$ is a diagonal matrix with its $(i, i)$-element equal to the sum of the $i$-th row of $W$.

3. Iterate $F(t+1) = \alpha SF(t) + (1-\alpha)Y$ until convergence, where $\alpha$ is a parameter in $(0, 1)$.

4. Let $F^*$ denote the limit of the sequence $\{F(t)\}$. Label each point $x_i$ as a label $y_i = \arg\max_{j \leq c} F_{ij}^*$.

This algorithm can be understood intuitively in terms of spreading activation networks [1, 11] from experimental psychology. We first define a pairwise relationship $W$ on the dataset $\mathcal{X}$ with the diagonal elements being zero. We can think that a graph $G = (V, E)$ is defined on $\mathcal{X}$, where the the vertex set $V$ is just $\mathcal{X}$ and the edges $E$ are weighted by $W$. In the second step, the weight matrix $W$ of $G$ is normalized symmetrically, which is necessary for the convergence of the following iteration. The first two steps are exactly the same as in spectral clustering [9]. During each iteration of the third step each point receives the information from its neighbors (first term), and also retains its initial information (second term). The parameter $\alpha$ specifies the relative amount of the information from its neighbors and its initial label information. It is worth mentioning that *self-reinforcement* is avoided since the diagonal elements of the affinity matrix are set to zero in the first step. Moreover, the information is spread *symmetrically* since $S$ is a symmetric matrix. Finally, the label of each unlabeled point is set to be the class of which it has received most information during the iteration process.

Let us show that the sequence $\{F(t)\}$ converges and $F^* = (1-\alpha)(I-\alpha S)^{-1}Y$. Without loss of generality, suppose $F(0) = Y$. By the iteration equation $F(t+1) = \alpha SF(t) + (1-\alpha)Y$ used in the algorithm, we have

$$F(t) = (\alpha S)^{t-1}Y + (1-\alpha)\sum_{i=0}^{t-1}(\alpha S)^i Y. \tag{1}$$

Since $0 < \alpha < 1$ and the eigenvalues of $S$ in [-1, 1] (note that $S$ is similar to the stochastic matrix $P = D^{-1}W = D^{-1/2}SD^{1/2}$),

$$\lim_{t\to\infty}(\alpha S)^{t-1} = 0, \text{ and } \lim_{t\to\infty}\sum_{i=0}^{t-1}(\alpha S)^i = (I-\alpha S)^{-1}. \tag{2}$$

Hence

$$F^* = \lim_{t\to\infty}F(t) = (1-\alpha)(I-\alpha S)^{-1}Y,$$

for classification, which is clearly equivalent to

$$F^* = (I-\alpha S)^{-1}Y. \tag{3}$$

Now we can compute $F^*$ directly without iterations. This also shows that the iteration result does not depend on the initial value for the iteration. In addition, it is worth to notice that $(I - \alpha S)^{-1}$ is in fact a graph or diffusion kernel [7, 12].

Now we discuss some possible variants of this method. The simplest modification is to repeat the iteration after convergence, i.e. $F^* = (I-\alpha S)^{-1}\cdots(I-\alpha S)^{-1}Y = (I-\alpha S)^{-p}Y$, where $p$ is an arbitrary positive integer. In addition, since that $S$ is similar to $P$, we can consider to substitute $P$ for $S$ in the third step, and then the corresponding closed form is $F^* = (I-\alpha P)^{-1}Y$. It is also interesting to replace $S$ with $P^T$, the transpose of $P$. Then the classifying function is $F^* = (I - \alpha P^T)^{-1}Y$. It is not hard to see this is equivalent to $F^* = (D - \alpha W)^{-1}Y$. We will compare these variants with the original algorithm in the experiments.

## 3 Regularization Framework

Here we develop a regularization framework for the above iteration algorithm. The cost function associated with $F$ is defined to be

$$Q(F) = \frac{1}{2}\left( \sum_{i,j=1}^{n} W_{ij} \left\| \frac{1}{\sqrt{D_{ii}}} F_i - \frac{1}{\sqrt{D_{jj}}} F_j \right\|^2 + \mu \sum_{i=1}^{n} \|F_i - Y_i\|^2 \right), \qquad (4)$$

Where $\mu > 0$ is the regularization parameter. Then the classifying function is

$$F^* = \arg\min_{F \in \mathcal{F}} Q(F). \qquad (5)$$

The first term of the right-hand side in the cost function is the *smoothness constraint*, which means that a good classifying function should not change too much between nearby points. The second term is the *fitting constraint*, which means a good classifying function should not change too much from the initial label assignment. The trade-off between these two competing constraints is captured by a positive parameter $\mu$. Note that the fitting constraint contains labeled as well as unlabeled data.

We can understand the smoothness term as the sum of the local variations, i.e. the local changes of the function between nearby points. As we have mentioned, the points involving pairwise relationships can be be thought of as an undirected weighted graph, the weights of which represent the pairwise relationships. The local variation is then in fact measured on each edge. We do not simply define the local variation on an edge by the difference of the function values on the two ends of the edge. The smoothness term essentially splits the function value at each point among the edges attached to it before computing the local changes, and the value assigned to each edge is proportional to its weight.

Differentiating $Q(F)$ with respect to $F$, we have

$$\left. \frac{\partial Q}{\partial F} \right|_{F=F^*} = F^* - SF^* + \mu(F^* - Y) = 0,$$

which can be transformed into

$$F^* - \frac{1}{1+\mu} SF^* - \frac{\mu}{1+\mu} Y = 0.$$

Let us introduce two new variables,

$$\alpha = \frac{1}{1+\mu}, \text{ and } \beta = \frac{\mu}{1+\mu}.$$

Note that $\alpha + \beta = 1$. Then

$$(I - \alpha S)F^* = \beta Y,$$

Since $I - \alpha S$ is invertible, we have

$$F^* = \beta(I - \alpha S)^{-1} Y. \qquad (6)$$

which recovers the closed form expression of the above iteration algorithm.

Similarly we can develop the optimization frameworks for the variants $F^* = (I - \alpha P)^{-1} Y$ and $F^* = (D - \alpha W)^{-1} Y$. We omit the discussions due to lack of space.

## 4 Experiments

We used $k$-NN and one-vs-rest SVMs as baselines, and compared our method to its two variants: (1) $F^* = (I - \alpha P)^{-1} Y$; and (2) $F^* = (D - \alpha W)^{-1} Y$. We also compared to Zhu *et al.*'s harmonic Gaussian field method coupled with the Class Mass Normalization (CMN) [15], which is closely related to ours. To the best of our knowledge, there is no reliable approach for model selection if only very few labeled points are available. Hence we let all algorithms use their respective optimal parameters, except that the parameter $\alpha$ used in our methods and its variants was simply fixed at 0.99.

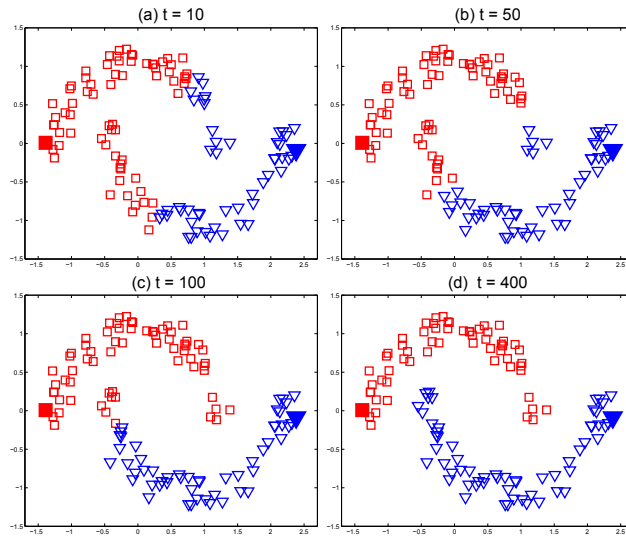

Figure 2: Classification on the pattern of two moons. The convergence process of our iteration algorithm with $t$ increasing from 1 to 400 is shown from (a) to (d). Note that the initial label information are diffused along the moons.

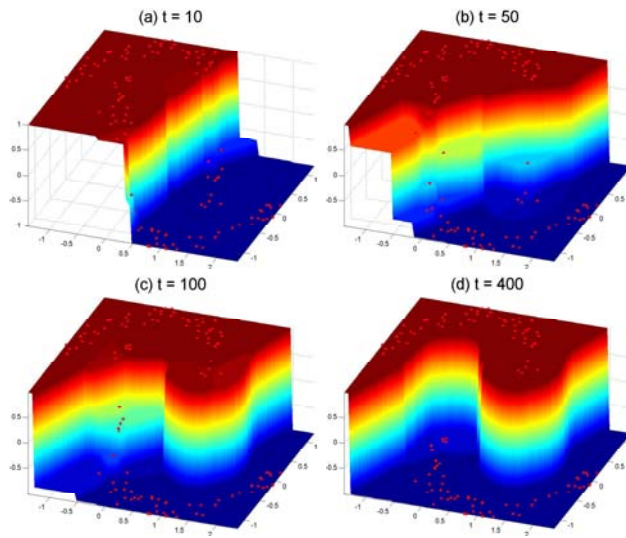

Figure 3: The real-valued classifying function becomes flatter and flatter with respect to the two moons pattern with increasing $t$. Note that two clear moons emerge in (d).

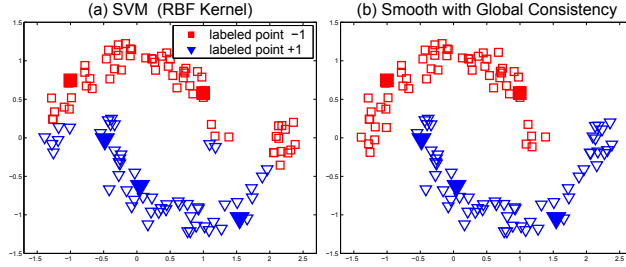

Figure 4: Smooth classification results given by supervised classifiers with the global consistency: (a) the classification result given by the SVM with a RBF kernel; (b) smooth the result of the SVM using the consistency method.

### 4.1 Toy Problem

In this experiment we considered the toy problem mentioned in Section 1 (Figure 1). The affinity matrix is defined by a RBF kernel but the diagonal elements are set to zero. The convergence process of our iteration algorithm with $t$ increasing from 1 to 400 is shown in Figure 2(a)-2(d). Note that the initial label information are diffused along the moons. The assumption of consistency essentially means that a good classifying function should change slowly on the coherent structure aggregated by a large amount of data. This can be illustrated by this toy problem very clearly. Let us define a function $f(x_i) = (F_{i1}^* - F_{i2}^*)/(F_{i1}^* + F_{i2}^*)$ and accordingly the decision function is $\text{sign}(f(x_i))$, which is equivalent to the decision rule described in Section 2. In Figure 3, we show that $f(x_i)$ becomes successively flatter with respect to the two moons pattern from Figure 3(a)-3(d) with increasing $t$. Note that two clear moons emerge in the Figure 3(d).

The basic idea of our method is to construct a smooth function. It is natural to consider using this method to improve a supervised classifier by smoothing its classifying result. In other words, we use the classifying result given by a supervised classifier as the input of our algorithm. This conjecture is demonstrated by a toy problem in Figure 4. Figure 4(a) is the classification result given by the SVM with a RBF kernel. This result is then assigned to $Y$ in our method. The output of our method is shown in Figure 4(b). Note that the points classified incorrectly by the SVM are successfully *smoothed* by the consistency method.

### 4.2 Digit Recognition

In this experiment, we addressed a classification task using the USPS handwritten 16x16 digits dataset. We used digits 1, 2, 3, and 4 in our experiments as the four classes. There are 1269, 929, 824, and 852 examples for each class, for a total of 3874.

The $k$ in $k$-NN was set to 1. The width of the RBF kernel for SVM was set to 5, and for the harmonic Gaussian field method it was set to 1.25. In our method and its variants, the affinity matrix was constructed by the RBF kernel with the same width used as in the harmonic Gaussian method, but the diagonal elements were set to 0. The test errors averaged over 100 trials are summarized in the left panel of Figure 5. Samples were chosen so that they contain at least one labeled point for each class. Our consistency method and one of its variant are clearly superior to the orthodox supervised learning algorithms $k$-NN and SVM, and also better than the harmonic Gaussian method.

Note that our approach does not require the affinity matrix $W$ to be positive definite. This enables us to incorporate prior knowledge about digit image invariance in an elegant way, e.g., by using a *jittered kernel* to compute the affinity matrix [5]. Other kernel methods are

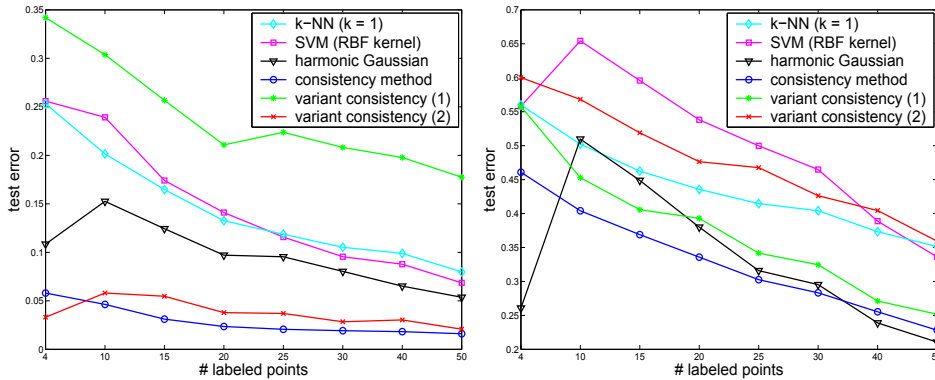

Figure 5: Left panel: the error rates of digit recognition with USPS handwritten 16x16 digits dataset for a total of 3874 (a subset containing digits from 1 to 4). Right panel: the error rates of text classification with 3970 document vectors in a 8014-dimensional space. Samples are chosen so that they contain at least one labeled point for each class.

known to have problems with this method [5]. In our case, jittering by 1 pixel translation leads to an error rate around 0.01 for 30 labeled points.

### 4.3 Text Classification

In this experiment, we investigated the task of text classification using the 20-newsgroups dataset. We chose the topic *rec* which contains *autos, motorcycles, baseball,* and *hockey* from the version 20-news-18828. The articles were processed by the Rainbow software package with the following options: (1) passing all words through the Porter stemmer before counting them; (2) tossing out any token which is on the stoplist of the SMART system; (3) skipping any headers; (4) ignoring words that occur in 5 or fewer documents. No further preprocessing was done. Removing the empty documents, we obtained 3970 document vectors in a 8014-dimensional space. Finally the documents were normalized into TFIDF representation.

The distance between points $x_i$ and $x_j$ was defined to be $d(x_i, x_j) = 1 - \langle x_i, x_j \rangle / \|x_i\| \|x_j\|$ [15]. The $k$ in $k$-NN was set to $1$. The width of the RBF kernel for SVM was set to $1.5$, and for the harmonic Gaussian method it was set to $0.15$. In our methods, the affinity matrix was constructed by the RBF kernel with the same width used as in the harmonic Gaussian method, but the diagonal elements were set to 0. The test errors averaged over 100 trials are summarized in the right panel of Figure 5. Samples were chosen so that they contain at least one labeled point for each class.

It is interesting to note that the harmonic method is very good when the number of labeled points is 4, i.e. one labeled point for each class. We think this is because there are almost equal proportions of different classes in the dataset, and so with four labeled points, the proportions happen to be estimated exactly. The harmonic method becomes worse, however, if slightly more labeled points are used, for instance, 10 labeled points, which leads to pretty poor estimation. As the number of labeled points increases further, the harmonic method works well again and somewhat better than our method, since the proportions of classes are estimated successfully again. However, our decision rule is much simpler, which in fact corresponds to the so-called *naive threshold*, the baseline of the harmonic method.

# 5 Conclusion

The key to semi-supervised learning problems is the consistency assumption, which essentially requires a classifying function to be sufficiently *smooth* with respect to the intrinsic structure revealed by a huge amount of labeled and unlabeled points. We proposed a simple algorithm to obtain such a solution, which demonstrated effective use of unlabeled data in experiments including toy data, digit recognition and text categorization. In our further research, we will focus on model selection and theoretic analysis.

### Acknowledgments

We would like to thank Vladimir Vapnik, Olivier Chapelle, Arthur Gretton, and Andre Elisseeff for their help with this work. We also thank Andrew Ng for helpful discussions about spectral clustering, and the anonymous reviewers for their constructive comments. Special thanks go to Xiaojin Zhu, Zoubin Ghahramani, and John Lafferty who communicated with us on the important post-processing step class mass normalization used in their method and also provided us with their detailed experimental data.

## References

[1] J. R. Anderson. *The architecture of cognition*. Harvard Univ. press, Cambridge, MA, 1983.

[2] M. Belkin and P. Niyogi. Semi-supervised learning on manifolds. *Machine Learning Journal*, to appear.

[3] A. Blum and S. Chawla. Learning from labeled and unlabeled data using graph mincuts. In *ICML*, 2001.

[4] O. Chapelle, J. Weston, and B. Schölkopf. Cluster kernels for semi-supervised learning. In *NIPS*, 2002.

[5] D. DeCoste and B. Schölkopf. Training invariant support vector machines. *Machine Learning*, 46:161–190, 2002.

[6] T. Joachims. Transductive learning via spectral graph partitioning. In *ICML*, 2003.

[7] J. Kandola, J. Shawe-Taylor, and N. Cristianini. Learning semantic similarity. In *NIPS*, 2002.

[8] R. I. Kondor and J. Lafferty. Diffusion kernels on graphs and other discrete input spaces. In *ICML*, 2002.

[9] A. Y. Ng, M. I. Jordan, and Y. Weiss. On spectral clustering: analysis and an algorithm. In *NIPS*, 2001.

[10] M. Seeger. Learning with labeled and unlabeled data. Technical report, The University of Edinburgh, 2000.

[11] J. Shrager, T. Hogg, and B. A. Huberman. Observation of phase transitions in spreading activation networks. *Science*, 236:1092–1094, 1987.

[12] A. Smola and R. I. Kondor. Kernels and regularization on graphs. In *Learning Theory and Kernel Machines*, Berlin - Heidelberg, Germany, 2003. Springer Verlag.

[13] M. Szummer and T. Jaakkola. Partially labeled classification with markov random walks. In *NIPS*, 2001.

[14] V. N. Vapnik. *Statistical learning theory*. Wiley, NY, 1998.

[15] X. Zhu, Z. Ghahramani, and J. Lafferty. Semi-supervised learning using gaussian fields and harmonic functions. In *ICML*, 2003.
